# Generic Analog Neural Computation — The EPSILON Chip

**Stephen Churcher**
Dept. of Elec. Engineering
University of Edinburgh
King's Buildings
Edinburgh, EH9 3JL

**Donald J. Baxter**
Dept. of Elec. Engineering
University of Edinburgh
King's Buildings
Edinburgh, EH9 3JL

**Alister Hamilton**
Dept. of Elec. Engineering
University of Edinburgh
King's Buildings
Edinburgh, EH9 3JL

**Alan F. Murray**
as above

**H. Martin Reekie**
as above

## Abstract

An analog CMOS VLSI neural processing chip has been designed and fabricated. The device employs "pulse-stream" neural state signalling, and is capable of computing some 360 million synaptic connections per second. In addition to basic characterisation results, the performance of the chip in solving "real-world" problems is also demonstrated.

## 1 INTRODUCTION

Inspired by biology, and borne out of a desire to perform analogue computation with fundamentally *digital* fabrication processes, the so-called "pulse-stream" arithmetic system has been steadily evolved and improved since its inception in 1986 (Murray1990a, Murray1989a). In addition to this continuous development at Edinburgh, many other research groups around the world (most notably Meador *et al* (Meador1990a)) have experimented with their own pulse-firing neural circuits.

In pulsed implementations, each neural state is represented by some variable attribute (e.g. the width of fixed frequency pulses, or the rate of fixed width pulses) of a train (or "stream") of pulses. The neuron design therefore reduces to a form of oscillator. Each neuron is fed by a column of synapses, which multiply incoming neural states by the synaptic weights. In contrast with the original circuits of Murray and Smith (Murray1987a), the synapse design which will be discussed herein utilises *analog* circuit techniques to perform the multiplication of neural state by synaptic weight.

This paper describes the Edinburgh Pulse-Stream Implementation of a Learning Oriented Network (EPSILON) chip. EPSILON was developed as a flexible neural processor, capable of addressing a variety of applications. The main design criteria were as follows :

• That it be large enough to be of use in practical problems.

• It should be capable of implementing networks of arbitrary size and architecture.

• It must be able to act as both a "slave" accelerator to a conventional computer, *and* as an "autonomous" processor.

As will be seen, these constraints resulted in a chip which could realise only a single layer of synaptic weights, but which could be *cascaded* to form large, useful networks for solving real-world problems.

The remaining sections of this paper describe the attributes of pulse-coded neural systems in general, before detailing the circuits which were employed on EPSILON. Finally, results from a vowel recognition application are presented, in order to illustrate the performance of EPSILON when applied to real tasks.

## 2   PULSE CODED NEURAL SYSTEMS

As already mentioned, EPSILON is a pulse coded analog neural processing chip. In such implementations, neural states are encoded as *digital* pulses. The states themselves may then be represented either by varying the *width* of the pulses (pulse width modulation — PWM), or by varying the *rate* of the pulses (pulse frequency modulation — PFM). The arguments for using pulses in this way are strong. Firstly, they provide a very effective and robust method for communicating states both on- and between-chip, since pulses are extremely resistant to noise. Secondly, the use of pulses to represent states renders interfacing to digital circuits and computer peripherals straightforward. Finally, pulsed signalling leads to simplification of artihmetic circuits (i.e. synapses), resulting in much higher inter-connection densities.

Unfortunately, pulse-based systems do have drawbacks. In common with all analog circuits, the synaptic computing elements have limited precision (usually equivalent to about 7 bits), and their performance is subject to the vagaries of fabrication process variations. This results in a situation whereby supposedly "matched" circuits vary markedly in their characteristics. Furthermore, the switching which is inherent in any pulsed circuit results in increased levels of system noise, most usually in the form of power supply transients. An additional problem with pulse frequency modulation (PFM) systems is that computation rates are dependent on the data; this is an important consideration in speed-critical applications.

## 3   THE EPSILON DESIGN

This section describes the circuits which were used in the EPSILON design. The operating principles of each circuit are discussed, and characterisation results presented. In accordance with the demerits mentioned in the previous section, all circuits were designed to be tolerant to noise and process variations, to *cause* as little noise as possible themselves, and to be easy to "set up" in practice. Finally, the specification of the EPSILON chip is presented.

### 3.1   SYNAPSE

The synapse design was based on the standard transconductance multiplier circuit, which had previously been the basis for monolithic analogue transversal filters for use in signal processing applications (Denyer1981a). Such multipliers use MOS transistors in their *linear* region of operation to generate output currents proportional to a product of two input voltages. This concept was adapted for use in pulsed neural networks by fixing one of the input voltages, and using a neural state to gate the output current. In this manner, the synaptic weight controls the magnitude of the output current, which is multiplied by the incoming neural pulses. The resultant charge packets are subsequently integrated to yield the total post-synaptic activity voltage.

Figure 1 shows the basic pulsed multiplier cell, where M1 and M2 form the transconductance multiplier, and M3 is the output pulse transistor. By ensuring that the drain-source voltages for M1 and M2 are the same and constant (the differential amplifier and transistors M4 and M5 are used to satisfy this constraint), non-linearities in the transistor responses can be cancelled out, such that $I_{OUT}$ is linearly dependent on the difference of $V_{GS1}$ and $V_{GS2}$ (Murray1992a). Multiplication is achieved by pulsing this current by the neural state, $V_j$. An "instantaneous" representation of the aggregated post-synaptic activity is given by the output voltage, $V_{OUT}$; this must subsequently be integrated in order to provide an activity input to a neuron.

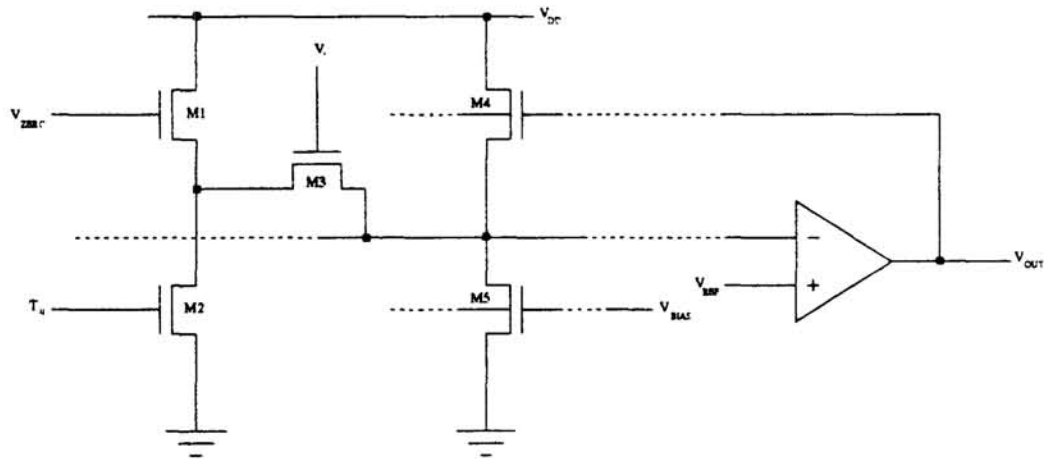

Figure 1: Transconductance Multiplier Synapse

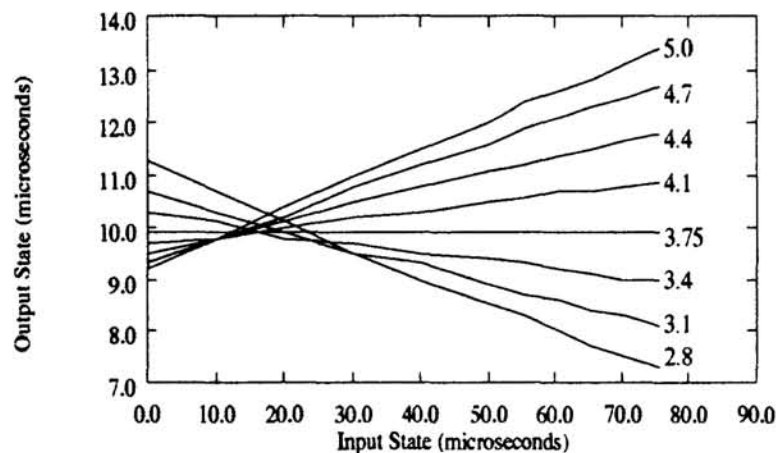

Figure 2: Synapse Characterisation Results

Results from characterisation tests of the synapse are presented in Figure 2, which shows output state against input state, for different synaptic weight voltages. As seen from the Figure, the linearity of the synapses, with respect to input state, is very high. The variation of synapse response with synaptic weight voltage is also fairly uniform. The graphs depict mean performance over all the synaptic columns in all the chips tested. The associated standard deviations were more or less constant, representing a variation of approximately ± 300 *ns* in the values of the output pulse widths. The effects of intra- and inter-chip process mismatches would therefore seem to be well contained by the circuit design. The "zero point" in the synaptic weight range was set at 3.75 *V* and, as can be seen from the Figure, each graph shows an offset problem when the input neural state is zero. This was attributable to an imbalance in the operating conditions of the transistors in the synapse, induced by the non-ideal nature of the power supplies (i.e. the non-zero sheet resistance of the power supply tracks), resulting in an offset in the input voltage to the post-synaptic integrator. This problem is easily obviated in practice, by employing three synapses per column to cancel the offset.

## 3.2   NEURONS

In order to reflect the diversity of neural network forms, and possible applications, two different neuron designs were included on the EPSILON chip. The first, a *synchronous* pulse width modulation neuron was designed with vision applications in mind. This circuit could guarantee network computation times, thereby eliminating the data dependency inherent in pulse frequency systems. The second neuron design used *asynchronous* pulse frequency modulation; the asynchronous nature of these circuits is advantageous in feedback and recurrent neural architectures, where temporal characteristics are important. As with the synapse, both circuits were designed to minimise transient noise injection, and to be tolerant of process variations.

### 3.2.1   Pulse Width Modulation

As already stated, this system retains all the advantages of using pulses for communication/calculation, whilst being able to *guarantee* a maximum network evaluation time. In the first instance, the main disadvantage with this technique appeared to be its synchronous nature - neurons would all be switching together causing larger power supply transients than in an asynchronous system. This problem has, however, been circumvented via a "double-sided" pulse modulation scheme, which will be more fully explained later.

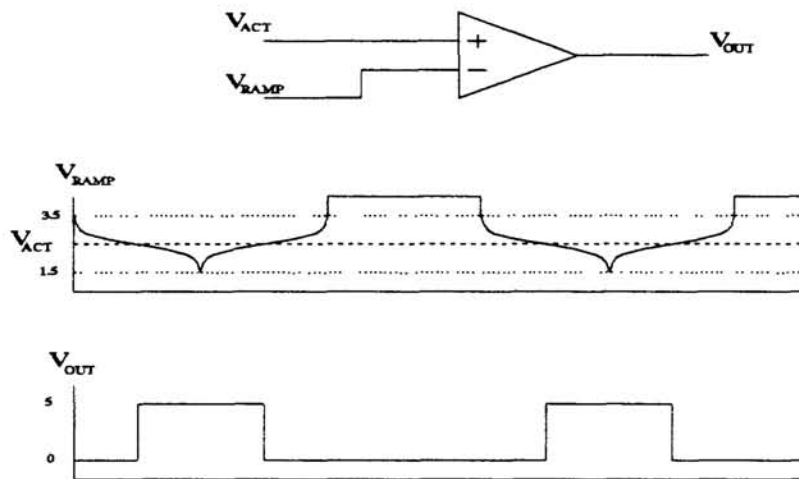

Figure 3:  Pulse-Width Modulation Neuron

The operation of the pulse-width modulation neuron is illustrated in Figure 3. The neuron itself is nothing more elaborate than a 2-stage comparator, with an inverter output driver stage. The inputs to the circuit are the integrated post-synaptic activity voltage, $V_{ACT}$, and a reference voltage, $V_{RAMP}$, which is generated off-chip and is globally distributed to all neurons in parallel. As seen from the waveforms in Figure 3, the output of the neuron changes state whenever the reference signal crosses the activity voltage level. An output pulse, which is some function of the input activation, is thus generated. The transfer function is entirely dependent on the shape of the reference signal - when this is generated by a RAM look-up table, the function can become completely arbitrary and hence user programmable. Figure 3 shows the signal which should be applied if a *sigmoidal* transfer characteristic is desired. Note that the sigmoid signals are "on their sides" - this is because the input (or independent variable) is on the vertical axis rather than the horizontal axis, as would normally be expected. The use of a "double-sided" ramp for the reference signal was alluded to earlier - this mechanism generates a pulse which is symmetrical about the mid-point of the ramp, thereby greatly reducing the likelihood of coincident

edges. This edge-asynchronicity obviates the problem of larger switching transients on the power supplies. Furthermore, because the analogue element (i.e. the ramp voltage) is effectively removed from the chip, and the circuit itself merely functions as a digital block, the system is immune to process variations.

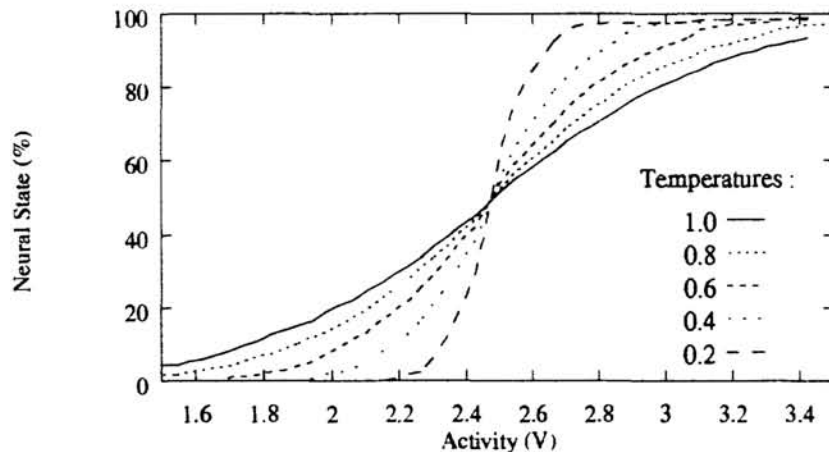

Figure 4: PWM Neuron Performance

Figure 4 shows plots of output state (measured as a percentage of a maximum possible 20 $\mu s$ pulse) versus input activity voltage, for five different sigmoid "temperatures", averaged over all the neurons on one chip. As can be seen, the fidelity of the sigmoids is extremely high, and it should be noted that all the curves are symmetrical about their mid-points — something which is difficult to achieve using standard analog circuits.

### 3.2.2    Pulse Frequency Modulation

The second neuron design which was included in EPSILON used pulse frequency encoding of the neural state. Although hampered by data dependent calculation times, its wholly asynchronous nature makes it ideal for neural network architectures which embody temporal characteristics i.e. feedback networks, and recurrent networks.

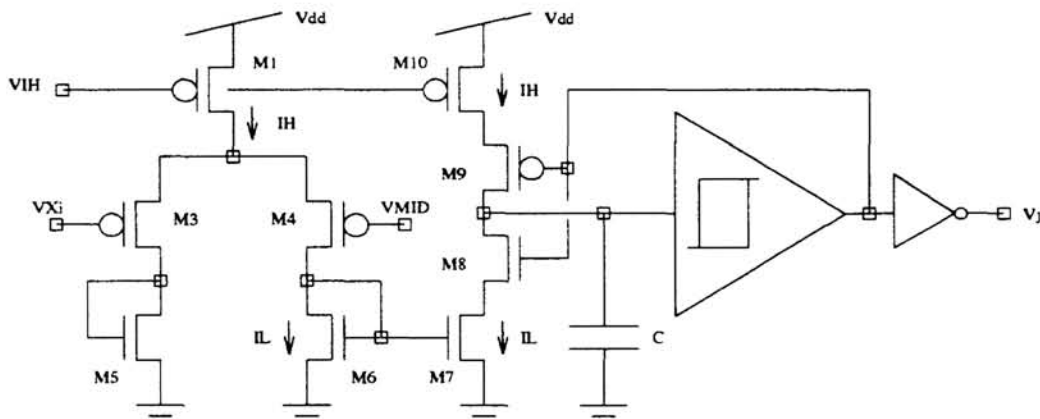

Figure 5: Pulse Frequency Modulation Neuron

The neuron design is illustrated in Figure 5, and is basically a Voltage Controlled Oscillator (VCO) with a variable gain sigmoidal transfer characteristic. Oscillation is achieved via the hysteretic charge and discharge of capacitor $C$, by the currents $IH$ and $IL$ respectively. The output pulse width is constant, and is set by $IH$, whilst the inter-pulse spacing (and hence output frequency) is controlled by $IL$. $IL$ itself is determined by the activity voltage, $VXI$, via the differential stage constituted by transistors M3 to M6. It is this latter which gives the VCO its sigmoidal characteristic, and gain variations may be achieved by injecting and removing additional current at appropriate points in this stage (note that the circuitry for this has been omitted from Figure 5, for the sake of clarity).

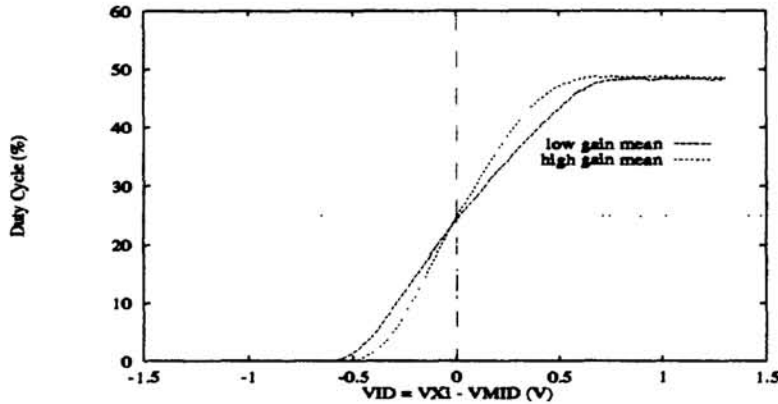

Figure 6: PFM Neuron Performance

The characterisation results for the VCO are presented in Figure 6. The Figure shows plots of output percentage duty cycle versus input differential voltage, for different values of sigmoid gain. Note that the curves are fair approximations to sigmoids, although, in contrast with the pulse width modulation neuron, they are *not* symmetrical about their mid-points. It can also be seen that the range of possible sigmoid gains is smaller than the range available with the PWM system, although this is not a crucial factor in many applications.

### 3.3    EPSILON SPECIFICATION

The circuits described in the previous section were combined to form the EPSILON chip. This was subsequently fabricated by European Silicon Structures (ES2) using their ECPD15 (i.e. 1.5 $\mu$m, double metal, single poly CMOS). As already stated, each chip was capable of implementing a single layer of synaptic connections, and could accept inputs as either analog voltages (for direct interface to sensors) or as pulses (for communication with other chips, and with digital systems). The full specification is given in Table 1.

## 4   APPLICATION — VOWEL RECOGNITION

After the device characterisation experiments had been completed, EPSILON was used to implement a multi-layer perceptron (MLP) for speech data classification. The MLP had 54 inputs, 27 hidden units, and 11 outputs, and the task was to classify 11 different vowel sounds spoken by each of 33 speakers. The input vectors were formed by the analog outputs of 54 band-pass filters.

The MLP was initially trained on a SPARC station, using a subset of 22 patterns. Learning (using the Virtual Targets algorithm, with 0 % noise (Murray1991a)) proceeded until the maximum bit error in the output vector was $\leq 0.3$, at which point the weight set was

Table 1: EPSILON Specifications

| EPSILON Specification | |
| --- | --- |
| No. of State Input Pins | 30 |
| No. of Actual State Inputs | 120, Muxed in Banks of 30 |
| Input Modes | Analog, PW, or PF |
| No. of State Outputs | 30, Directly Pinned Out |
| Output Modes | PW or PF |
| No. of Synapses | 3600 |
| No. of Weight Load Channels | 2 |
| Weight Load Time | $3.6\ ms$ |
| Weight Storage | Dynamic |
| Maximum Speed (cps) | $360\ Mcps$ |
| Technology | $1.5\ \mu m$, Double Metal CMOS |
| Die Size | $9.5\ mm \times 10.1\ mm$ |
| Maximum Power Dissipation | $350\ mW$ |

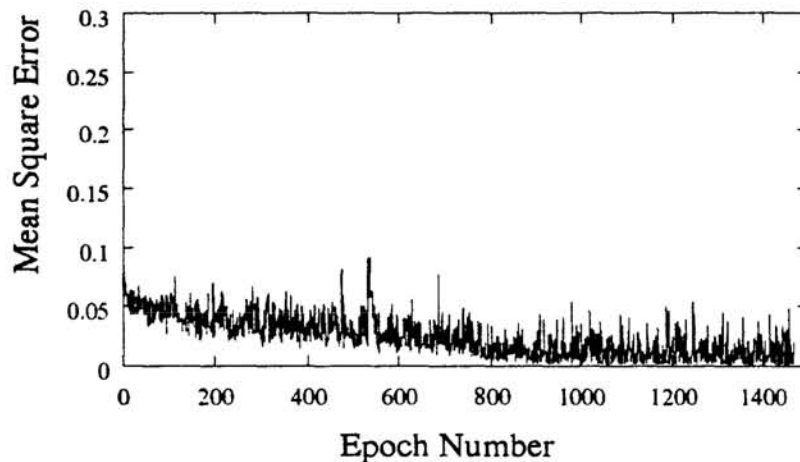

Figure 7: EPSILON Under Training

downloaded to EPSILON. Training was then restarted under the same regime as before (using the same "stop" criterion), although this time EPSILON was used to evaluate the "forward pass" phases of the network. Figure 7 shows the evolution of mean square error with number of epochs during this period; at the end of training, EPSILON could correctly identify all 22 training patterns.

Subsequent to this, 176 "unseen" test patterns were presented to the EPSILON network, with the result that 65.34 % of these vectors were correctly classified. This compared very favourably with similar generalisation experiments which were carried out on a SPARC : in this case, the best result obtained was 67.61 %.

## 5 CONCLUSIONS

In conclusion, a large analog VLSI neural chip, composed of process tolerant circuits, with useful characteristics has been fabricated. Although not a *self*-learning device, it has been proved that EPSILON will support learning, and can be applied successfully to real-

world problems. Indeed, when correctly trained, the performance of the EPSILON chip has been shown to be comparable with that of software simulations on a SPARC station.

Work is currently under way to apply EPSILON to computer vision tasks; more specifically, it it will be used to implement an MLP which is capable of recognising regions in segmented images of natural scenes. Furthermore, the success of the learning experiments has given us sufficient confidence to undertake the development of a self-learning analog neural chip. It is envisaged that this will employ EPSILON-type circuits, and will implement the Virtual Targets (Murray1991a) training algorithm; the design of a small prototype is currently nearing completion.

## Acknowledgements

The authors would like to thank the Science and Engineering Research Council for their continued funding of this work. In addition, Stephen Churcher and Donald Baxter are grateful to British Aerospace PLC and Thorn-EMI CRL respectively for sponsorship and technical support during the course of their PhD's.

Lionel Tarassenko (Dept. of Engineering Science, University of Oxford) must also be thanked for his invaluable comments, and for supplying the vowel database.

## References

Murray1990a.
A. F. Murray, D. Baxter, Z. Butler, S. Churcher, A. Hamilton, H. M. Reekie, and L. Tarassenko, "Innovations in Pulse Stream Neural VLSI : Arithmetic and Communications", *IEEE Workshop on Microelectronics for Neural Networks, Dortmund 1990*, pp. 8-15, 1990.

Murray1989a.
A. F. Murray, "Pulse Arithmetic in VLSI Neural Networks", *IEEE MICRO*, vol. 9, no. 6, pp. 64-74, 1989.

Meador1990a.
J. Meador, A. Wu, C. Cole, N. Nintunze, and P. Chintrakulchai, "Programmable Impulse Neural Circuits", *IEEE Transactions on Neural Networks*, vol. 2, no. 1, pp. 101-109, 1990.

Murray1987a.
A. F. Murray and A. V. W. Smith, "Asynchronous Arithmetic for VLSI Neural Systems", *Electronics Letters*, vol. 23, no. 12, pp. 642-643, June, 1987.

Denyer1981a.
P. B. Denyer and J. Mavor, "MOST Transconductance Multipliers for Array Applications", *IEE Proc. Pt. 1*, vol. 128, no. 3, pp. 81-86, June 1981.

Murray1992a.
A.F. Murray, A. Hamilton, D.J. Baxter, S. Churcher, H.M. Reekie, and L. Tarassenko, "Integrated Pulse-Stream Neural Networks - Results, Issues and Pointers", *IEEE Trans. Neural Networks*, pp. 385-393, 1992.

Murray1991a.
A. F. Murray, "Analog VLSI and Multi-Layer Perceptrons - Accuracy, Noise and On-Chip Learning", *Proc. Second International Conference on Microelectronics for Neural Networks, Munich (Germany)*, pp. 27-34, 1991.